# Unifying the Sensory and Motor Components of Sensorimotor Adaptation

**Adrian Haith**
School of Informatics
University of Edinburgh, UK
adrian.haith@ed.ac.uk

**Carl Jackson**
School of Psychology
University of Birmingham, UK
c.p.jackson.1@bham.ac.uk

**Chris Miall**
School of Psychology
University of Birmingham, UK
r.c.miall@bham.ac.uk

**Sethu Vijayakumar**
School of Informatics
University of Edinburgh, UK
sethu.vijayakumar@ed.ac.uk

## Abstract

Adaptation of visually guided reaching movements in novel visuomotor environments (e.g. wearing prism goggles) comprises not only motor adaptation but also substantial sensory adaptation, corresponding to shifts in the perceived spatial location of visual and proprioceptive cues. Previous computational models of the sensory component of visuomotor adaptation have assumed that it is driven purely by the discrepancy introduced between visual and proprioceptive estimates of hand position and is independent of any motor component of adaptation. We instead propose a unified model in which sensory and motor adaptation are jointly driven by optimal Bayesian estimation of the sensory and motor contributions to perceived errors. Our model is able to account for patterns of performance errors during visuomotor adaptation as well as the subsequent perceptual aftereffects. This unified model also makes the surprising prediction that force field adaptation will elicit similar perceptual shifts, even though there is never any discrepancy between visual and proprioceptive observations. We confirm this prediction with an experiment.

## 1 Introduction

When exposed to a novel visuomotor environment, for instance while wearing prism goggles, subjects initially exhibit large directional errors during reaching movements but are able to rapidly adapt their movement patterns and approach baseline performance levels within around 30-50 reach trials. Such *visuomotor adaptation* is multifaceted, comprising both sensory and motor components [5]. The sensory components of adaptation can be measured through alignment tests in which subjects are asked to localize either a visual target or their unseen fingertip, with their other (also unseen) fingertip (without being able to make contact between hands). These tests reveal substantial shifts in the perceived spatial location of both visual and proprioceptive cues, following adaptation to shifted visual feedback [7].

While a shift in visual spatial perception will be partially reflected in reaches towards visual targets, sensory adaptation alone cannot fully account for the completenes of visuomotor adaptation, since the shifts in visual perception are always substantially less than the experimentally-imposed shift. There must therefore be some additional motor component of adaptation, i.e. some change in the relationship between the planned movement and the

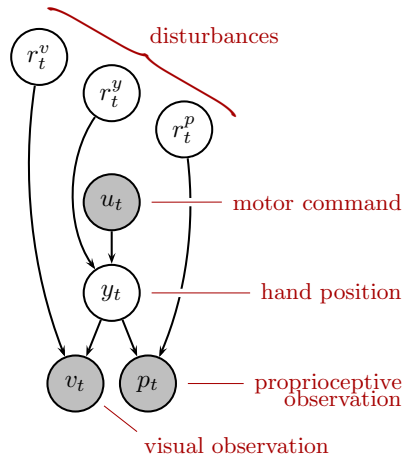

Figure 1: Graphical model of a single reach in a motor adaptation experiment. Motor command $u_t$, and visual and proprioceptive observations of hand position, $v_t$ and $p_t$, are available to the subject. Three distinct disturbances affect observations: A motor disturbance $r_t^y$ may affect the hand position $y_t$ given the motor command $u_t$. Visual and proprioceptive disturbances, $r_t^v$ and $r_t^p$, may affect the respective observations given hand position.

issued motor command. This argument is reinforced by the finding that patterns of reach aftereffects following visuomotor adaptation depend strongly on the motor task performed during adaptation [5].

From a modelling point of view, the sensory and motor components of adaptation have previously only been addressed in isolation of one another. Previously proposed models of sensory adaptation have assumed that it is driven purely by discrepancies between hand position estimates from different sensory modalities. Ghahramani et al. [2] proposed a computational model based on a maximum likelihood principle, details of which we give in Section 3. On its own, this sensory adaptation model cannot provide a complete description of visuomotor adaptation since it does not fully account for improvements in performance from trial to trial. It can, however, be plausibly combined with a conventional error-driven motor adaptation model in which the performance error is calculated using the maximum likelihood estimate of hand position. The resulting composite model could plausibly account for both performance improvements and perceptual shifts during visuomotor adaptation. According to this view, sensory and motor adaptation are very much independent processes, one driven by sensory discrepancy and the other driven by (estimated) task performance error.

In Section 4, we argue for a more unified view of sensory and motor adaptation in which all three components of adaptation are jointly guided by optimal Bayesian inference of the corresponding potential sources of error experienced on each trial, given noisy visual and proprioceptive observations of performance and noisy motor execution. This unified sensory and motor adaptation model is also able to account for both performance improvements and perceptual shifts during visuomotor adaptation. However, our unified model also makes the surprising prediction that a motor disturbance, e.g. an external force applied to hand via a manipulandum, will also elicit sensory adaptation. The MLE-based model predicts no such sensory adaptation, since there is never any discrepancy between sensory modalities. We test this prediction directly with an experiment (Section 5) and find that force field adaptation does indeed lead to sensory as well as motor adaptation.

## 2 Modelling framework

Before describing the details of the models, we first outline a basic mathematical framework for describing reaching movements in the context of a motor adaptation experiment, representing the assumptions common to both the MLE-based and the Bayesian adaptation models. Figure 1 illustrates a graphical model of a single reaching movement during an adaptation experiment, from the subject's point of view. The multiple components of visuomotor adaptation described above correspond to three distinct potential sources of observed outcome error (across both observation) modalities in a single reaching trial.

On trial $t$, the subject generates a (known) motor command $u_t$. This motor command $u_t$ leads to a final hand position $y_t$, which also depends on some (unknown) motor disturbance

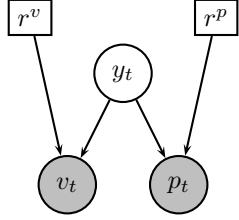

Figure 2: MLE-based sensor adaptation model. Visual and proprioceptive disturbances $r^v$, $r^p$ are treated as *parameters* of the model. Estimates $\hat{r}_t^v$ and $\hat{r}_t^p$ of these parameters are maintained via an online EM-like procedure.

$r_t^y$ (e.g. an external force applied to the hand) and motor noise $\epsilon_t^u$. We assume the final hand position $y_t$ is given by

$$y_t = u_t + r_t^y + \epsilon_t^u, \tag{1}$$

where $\epsilon_t^u \sim N(0, \sigma_u^2)$. Although this is a highly simplified description of the forward dynamics of the reaching movement, it can be regarded as a first-order approximation to the true dynamics. Similar assumptions have proved very successful elsewhere in models of force field adaptation, e.g. [1]

The experimenter ultimately measures the hand position $y_t$, however this is not directly observed by the subject. Instead, noisy and potentially shifted observations are available through visual and proprioceptive modalities,

$$v_t = y_t + r_t^v + \epsilon_t^v, \tag{2}$$
$$p_t = y_t + r_t^p + \epsilon_t^p, \tag{3}$$

where the observation noises $\epsilon_t^v$ and $\epsilon_t^p$ are zero-mean and Gaussian with variances $\sigma_v^2$ and $\sigma_p^2$, respectively.

We denote the full set of potential disturbances on trial $t$ by

$$\mathbf{r}_t = (r_t^v,\ r_t^p,\ r_t^y)^T. \tag{4}$$

We assume that the subject maintains an internal estimate $\hat{\mathbf{r}}_t = (\hat{r}_t^v, \hat{r}_t^p, \hat{r}_t^y)^T$ of the total disturbance $\mathbf{r}_t$ and selects his motor commands on each trial accordingly. For reaches to a visual target located at $v_t^*$, the appropriate motor command is given by

$$u_t = v_t^* - \hat{r}_t^v - \hat{r}_t^y. \tag{5}$$

Adaptation can be viewed as a process of iteratively updating the disturbance estimate, $\hat{\mathbf{r}}_t$, following each trial given the new (noisy) observations $v_t$ and $p_t$ and the motor command $u_t$. Exactly how the subject uses the information available to infer the current disturbances is the subject of subsequent sections of this paper.

## 3   Existing sensory adaptation models

The prevailing view of sensory adaptation centres around the principle of maximum likelihood estimation and was first proposed by Ghahramani et al. [2] in the context of combining discrepant visual and auditory cues in a target location task. It has nevertheless been widely accepted as a model of how the nervous system deals with visual and proprioceptive cues. Van Beers et al. [7], for instance, based an analysis of the relative uncertainty of visual and proprioceptive estimates of hand location on this principle.

We suppose that, given the subject's current estimate of the visual and proprioceptive disturbance, $\hat{r}_t^v$ and $\hat{r}_t^p$, the visual and proprioceptive estimates of hand position are given by

$$\hat{y}_t^v = v_t - \hat{r}_t^v, \tag{6}$$
$$\hat{y}_t^p = p_t - \hat{r}_t^p \tag{7}$$

respectively. These distinct estimates of hand position are combined via maximum likelihood estimation [7] into a single fused estimate of hand position. The maximum likelihood estimate (MLE) of the true hand position $y_t$ is given by

$$\hat{y}_t^{MLE} = \frac{\sigma_p^2}{\sigma_v^2 + \sigma_p^2}\hat{y}_t^v + \frac{\sigma_v^2}{\sigma_v^2 + \sigma_p^2}\hat{y}_t^p. \tag{8}$$

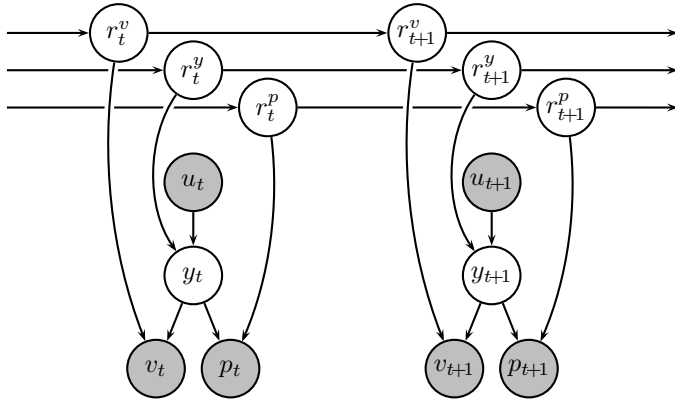

Figure 3: Bayesian combined sensory and motor adaptation model. The subject assumes that disturbances vary randomly, but smoothly, from trial to trial.

The MLE-based sensory adaptation model states that subjects adapt their future visual and proprioceptive estimates of hand location towards the MLE in such a way that the MLE itself remains unchanged. The corresponding updates are given by

$$\hat{r}^v_{t+1} = \hat{r}^v_t + \eta w_p \left[ \hat{y}^p_t - \hat{y}^v_t \right], \qquad (9)$$
$$\hat{r}^p_{t+1} = \hat{r}^p_t + \eta w_v \left[ \hat{y}^v_t - \hat{y}^p_t \right], \qquad (10)$$

where $\eta$ is some fixed adaptation rate. This adaptation principle can be interpreted as an online expectation-maximization (EM) procedure in the graphical model shown in Figure 2. In this model, $r^v$ and $r^p$ are treated as *parameters* of the model. The E-step of the EM procedure corresponds to finding the MLE of $y_t$ and the M-step corresponds to gradient ascent on the likelihood of $\hat{r}^v$ and $\hat{r}^p$.

## 3.1 Extending the MLE model to account for motor component of adaptation

As it stands, the MLE-based model described above only accounts for sensory adaptation and does not provide a complete description of sensorimotor adaptation. Visual adaptation will affect the estimated location of a visual target, and therefore also the planned movement, but the effect on performance will not be enough to account for complete (or nearly complete) adaptation. The performance gain from this component of adaptation will be equal to the discrepancy between the initial visual setimate of hand posion and the MLE - which will be substantially less than the experimentally imposed shift.

This sensory adaptation model can, however, be plausibly combined with a conventional error-driven state space model [6, 1] of motor adaptation to yield an additional motor component of adaptation $\hat{r}^y_t$. The hand position MLE $\hat{y}_t$ can be used in place of the usual uni-modal observation assumed in these models when calculating the endpoint error. The resulting update for the estimated motor disturbance $\hat{r}^y_t$ on trial $t$ is given by

$$\hat{r}^y_{t+1} = \hat{r}^y_t + \gamma(\hat{y}^*_t - \hat{y}^{MLE}_t), \qquad (11)$$

where $\hat{y}^*_t = (v^* - \hat{r}^v_t)$ is the estimated desired hand location, and $\gamma$ is some fixed adaptation rate.

This combined model reflects the view that sensory and motor adaptation are distinct processes. The sensory adaptation component is driven purely by discrepancy between the senses, while the motor adaptation component only has access to a single, fused estimate of hand position and is driven purely by estimated performance error.

## 4 Unified Bayesian sensory and motor adapatation model

We propose an alternative approach to solving the sensorimotor adaptation problem. Rather than treat the visual shifts $r^v$ and $r^p$ as parameters, we consider all the disturbances (including $r^y_t$) as dynamic random variables. We assume that the subject's beliefs about how

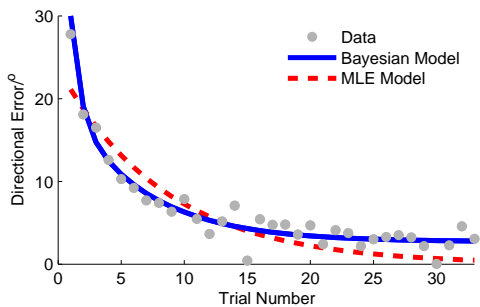

Figure 4: Model comparison with visuomotor adaptation data. The Bayesian model (solid blue line) and MLE-based model (dashed red line) were fitted to performance data (filled circles) from a visuomotor adaptation experiment [4]. Both models made qualitatively similar predictions about how adaptation was distributed across components.

these disturbances evolve over time are characterised by a trial-to-trial disturbance dynamics model given by

$$\mathbf{r}_{t+1} = A\mathbf{r}_t + \boldsymbol{\eta}_t, \tag{12}$$

where $A$ is some diagonal matrix and $\boldsymbol{\eta}_t$ is a random drift term with zero mean and diagonal covariance matrix $Q$, i.e.

$$\boldsymbol{\eta}_t \sim N(0, Q). \tag{13}$$

$A$ and $Q$ are both diagonal to reflect the fact that each disturbance evolves independently. We denote the diagonal elements of $A$ by $\mathbf{a} = (a^v, a^p, a^u)$ and the diagonal of $Q$ by $\mathbf{q} = (q^v, q^p, q^u)$. The vector $\mathbf{a}$ describes the timescales over which each disturbance persists, while $\mathbf{q}$ describes the amount of random variation from trial to trial, or volatility of each disturbance. These parameters reflect the statistics of the usual fluctuations in sensory calibration errors and motor plant dynamics, which the sensorimotor system must adapt to on an ongoing basis. (Similar assumptions have previously been made elsewhere [3, 4]).

Combining these assumptions with the statistical model of each individual trial described in Section 2 (and Figure 1), gives rise to a dynamical model of the disturbances and their impact on reaching movements, across all trials. This model, representing the subjects beliefs about how his sensorimotor performance is liable to vary over time, is illustrated in Figure 4. We propose that the patterns of adaptation and the sensory aftereffects exhibited by subjects correspond to optimal inference of the disturbances $\mathbf{r}_t$ within this model, given the observations on each trial.

The linear dynamics and Gaussian noise of the observer's model mean that exact inference is straightforward and equivalent to a Kalman filter. The latent state tracked by the Kalman filter is the vector of disturbances $\mathbf{r}_t = (r_t^v, r_t^p, r_t^y)^T$, with state dynamics given by (12). The observations $v_t$ and $p_t$ are related to the disturbances via

$$\begin{pmatrix} v_t \\ p_t \end{pmatrix} = \begin{pmatrix} u_t \\ u_t \end{pmatrix} + \begin{pmatrix} 1 & 0 & 1 \\ 0 & 1 & 1 \end{pmatrix} (\mathbf{r}_t + \boldsymbol{\epsilon}_t), \tag{14}$$

where $\boldsymbol{\epsilon}_t = (\epsilon_t^v, \epsilon_t^p, \epsilon_t^u)^T$. We can write this in a more conventional form as

$$\mathbf{z}_t = H\mathbf{r}_t + H\boldsymbol{\epsilon}_t, \tag{15}$$

where $\mathbf{z}_t = (v_t - u_t, p_t - u_t)^T$ and $H$ is the matrix of 1's and 0's in equation (14). The observation noise covariance is given by

$$R = E\left[(H\boldsymbol{\epsilon}_t)(H\boldsymbol{\epsilon}_t)^T\right] = \begin{pmatrix} \sigma_v^2 + \sigma_u^2 & \sigma_u^2 \\ \sigma_u^2 & \sigma_p^2 + \sigma_u^2 \end{pmatrix}. \tag{16}$$

The standard Kalman filter update equations can be used to predict how a subject will update estimates of the disturbances following each trial and therefore how he will select his actions on the next trial, leading to a full prediction of performance from the first trial onwards.

## 5   Model comparison and experiments

We have described two alternative models of visuomotor adaptation which we have claimed can account for both the motor and sensory components of adaptation. We fitted both

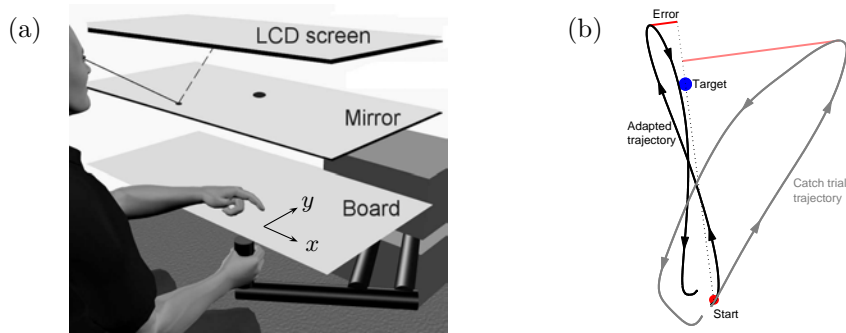

Figure 5: (a) Experimental Setup, (b) Sample trajectories and performance error measure

models to performance data from a visuomotor adaptation experiment [4] to validate this claim. In this study in which this data was taken from, subjects performed visually guided reaching movements to a number of targets. Visual feedback of hand position (given via a cursor on a screen) was rotated by $30^o$ relative to the starting position of each movement. The mean directional error (averaged over targets and over subjects) over trials is plotted in Figure 4. The Matlab function `lsqnonlin` was used to find the parameters for each model which minimized the sum of the error between the data and the predictions of each model. There were 5 free parameters for the MLE-based model $(\sigma_v^2, \sigma_p^2, \sigma_u^2, \eta, \gamma)$. For the Bayesian model we assumed that all disturbances had the same timescale, i.e. all elements of **a** were the same, leaving 7 free parameters $(\sigma_v^2, \sigma_p^2, \sigma_u^2, q^v, q^p, q^u, a)$. The results of the fits are shown in Figure 4. The spread of adaptation across components of the model was qualitatively similar between the two models, although no data on perceptual aftereffects was available from this study for quantitative comparison. The Bayesian model clearly displays a closer fit to the data and the Akaike information criterion (AIC) confirmed that this was not simply due to extra parameters ($AIC = 126.7$ for the Bayesian model vs $AIC = 159.6$ for the MLE-based model).

Although the Bayesian model appears to describe the data better, this analysis is by no means conclusive. Furthermore, the similar scope of predictions between the two models means that gathering additional data from alignment tests may not provide any further leverage to distinguish between the two models. There is, however, a more striking difference in predictions between the two models. While the MLE-based model predicts there will be sensory adaptation *only* when there is a discrepancy between the senses, the Bayesian model predicts that there will also be sensory adaptation in response to a motor disturbance such as an external force applied to the hand). Just as a purely visual disturbance can lead to a multifaceted adaptive response, so can a purely motor disturbance, with both motor and sensory components predicted, even though there is never any discrepancy between the senses. This prediction enables us to distinguish decisively between the two models.

## 5.1 Experimental Methods

We experimentally tested the hypothesis that force field adaptation would lead to sensory adaptation. We tested 11 subjects who performed a series of trials consisting of reaching movements interleaved with perceptual alignment tests.

Subjects grasped the handle of a robotic manipulandum with their right hand. The hand was not visible directly, but a cursor displayed via a mirror/flat screen monitor setup (Figure 5.1(a)) was exactly co-planar and aligned with the handle of the manipulandum. In the movement phase, subjects made an out-and-back reaching movement towards a visual target with their right hand. In the visual localization phase, a visual target was displayed pseudorandomly in one of 5 positions and the subjects moved their left fingertip to the perceived location of the target. In the proprioceptive localization phase, the right hand was passively moved to a random target location, with no visual cue of its position, and subjects moved their left fingertip to the perceived location of the right hand. Left fingertip

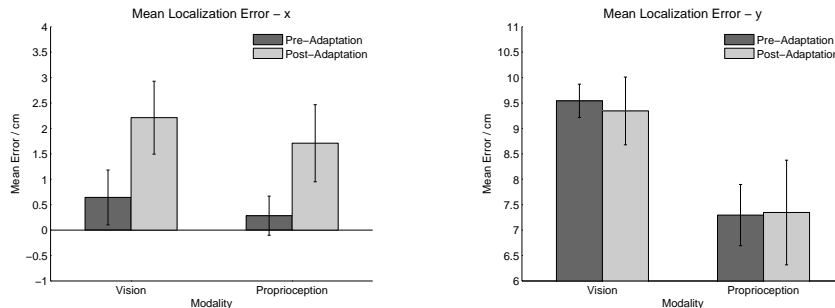

Figure 6: (a) Average lateral (in direction of the perturbation) localization error across subjects before vs after adaptation, for vision and proprioception. Error bars indicate standard errors. (b) Same plots for $y$-direction

positions were recorded using a Polhemus motion tracker. Neither hand was directly visible at any time during the experiment.

Subjects were given 25 baseline trials with zero external force, after which a force field was gradually introduced. A leftward lateral force $F_x$ was applied to the right hand during the reaching phase. The magnitude of the force was proportional to the forward velocity $\dot{y}$ of the hand, i.e.

$$F_x = -a\dot{y}. \tag{17}$$

The force was applied only on the outward part of the movement (i.e. only when $\dot{y} > 0$). After steadily incrementing $a$ during 50 adaptation trials, the force field was then kept constant at $a = 0.3 \ N/(cms^{-1})$ for a further 25 post-adaptation test trials. All subjects received a catch trial at the very end in which the force field was turned off.

The particular force field used was chosen so that the cursor trajectories (and motor commands required to counter the perturbation) would be as close as possible to those used to generate the linear trajectories required when exposed to a visuomotor shift (such as that described in [7]). Figure 5.1(b) shows two trajectories from a typical subject, one from the post-adaptation test phase and one from the catch trial after adaptation. The initial outward part of the catch trial trajectory, the initial movement is very straight, implying that similar motor commands were used to those required by a visuomotor shift.

## 5.2    Results

We compared the average performance in the visual and proprioceptive alignment tests before and after adaptation in the velocity-dependent force field. The results are summarized in Figure 6(a). Most subjects exhibited small but significant shifts in performance in both the visual and proprioceptive alignment tests. Two subjects exhibited shifts which were more than two standard deviations away from the average shift and were excluded from the analysis. We found significant lateral shifts in both visual and proprioceptive localization error in the direction of the perturbation (both p < .05, one-tailed paired t-test). Figure 6(b) shows the same data for the direction perpendicular to the perturbation. Although the initial localization bias was high, there was no significant shift in this direction following adaptation.

We quantified each subject's performance on each trial as the perpendicular distance of the furthest point in the trajectory from the straight line between the starting point and the target (Fig. 5.1(b)). We fitted the Bayesian and MLE-based models to the data following the same procedure as before, only this time penalizing the disagreement between the model and the data for the alignment tests, in addition to the reaching performance. Figure 7 illustrates the averaged data along with the model fits. Both models were able to account reasonably well for the trends in reaching performance across trials (7(a)). Figures 7(b) and 7(c) show the model fits for the perceptual localization task. The Bayesian model is able to account for both the extent of the shift and the timecourse of this shift during adaptation.

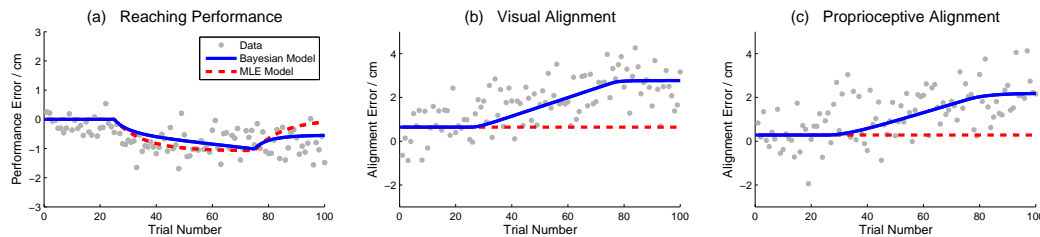

Figure 7: Trial-by-trial data and model fits. (a) Reaching error, (b) Visual alignment test error, (c) Proprioceptive alignment test error. The Bayesian (solid blue lines) and MLE-based (dashed red lines) were fitted to averaged data across subjects (circles).

Since there was never any sensory discrepancy, the MLE-based model predicted no change in the localization task.

## 6   Conclusions and discussion

Our experimental results demonstrate that adaptation of reaching movements in a force field results in shifts in visual and proprioceptive spatial perception. This novel finding strongly supports the Bayesian model, which predicted such adaptation, and refutes the MLE-based model, which did not. The Bayesian model was able to account for the trends in both reaching performance and alignment test errors on a trial-to-trial basis.

Several recent models have similarly described motor adaptation as a process of Bayesian inference of the potential causes of observed error. Körding et al. [3] proposed a model of saccade adaptation and Krakauer et al. [4] modelled visuomotor adaptation based on this principle. Our work extends the framework of these models to include multiple observation modalities instead of just one, and multiple classes of disturbances which affect the different observation modalities in different, experimentally measurable ways.

Overall, our results suggest that the nervous system solves the problems of sensory and motor adaptation in a principled and unified manner, supporting the view that sensorimotor adaptation proceeds according to optimal estimation of encountered disturbances.

## References

[1] Opher Donchin, Joseph T Francis, and Reza Shadmehr. Quantifying generalization from trial-by-trial behavior of adaptive systems that learn with basis functions: theory and experiments in human motor control. *J Neurosci*, 23(27):9032–9045, Oct 2003.

[2] Z. Ghahramani, D.M. Wolpert, and M.I. Jordan. Computational models for sensorimotor integration. In P.G. Morasso and V. Sanguineti, editors, *Self-Organization, Computational Maps and Motor Control*, pages 117–147. North-Holland, Amsterdam, 1997.

[3] Konrad P. Körding, Joshua B. Tenenbaum, and Reza Shadmehr. The dynamics of memory as a consequence of optimal adaptation to a changing body. *Nat Neurosci*, 10(6):779–786, June 2007.

[4] John W Krakauer, Pietro Mazzoni, Ali Ghazizadeh, Roshni Ravindran, and Reza Shadmehr. Generalization of motor learning depends on the history of prior action. *PLoS Biol*, 4(10):e316, Sep 2006.

[5] M.C. Simani, L.M. McGuire, and P.N. Sabes. Visual-shift adaptation is composed of separable sensory and task-dependent effects. *J Neurophysiol*, 98:2827–2841, Nov 2007.

[6] K A Thoroughman and R Shadmehr. Learning of action through adaptive combination of motor primitives. *Nature*, 407(6805):742–747, Oct 2000.

[7] Robert J van Beers, Daniel M Wolpert, and Patrick Haggard. When feeling is more important than seeing in sensorimotor adaptation. *Curr Biol*, 12(10):834–837, May 2002.
